# On Learning Rotations

**Raman Arora**
University of Wisconsin-Madison
Department of Electrical and Computer Engineering
1415 Engineering Drive, Madison, WI 53706
rmnarora@u.washington.edu

## Abstract

An algorithm is presented for online learning of rotations. The proposed algorithm involves matrix exponentiated gradient updates and is motivated by the von Neumann divergence. The multiplicative updates are exponentiated skew-symmetric matrices which comprise the Lie algebra of the rotation group. The orthonormality and unit determinant of the matrix parameter are preserved using matrix logarithms and exponentials and the algorithm lends itself to intuitive interpretation in terms of the differential geometry of the manifold associated with the rotation group. A complexity reduction result is presented that exploits the eigenstructure of the matrix updates to simplify matrix exponentiation to a quadratic form.

## 1  Introduction

The problem of learning rotations finds application in many areas of signal processing and machine learning. It is an important problem since many problems can be reduced to that of learning rotations; for instance Euclidean motion in $\mathbb{R}^{n-1}$ is simply rotation in $\mathbb{R}^n$. A conformal embedding was presented in [1] that extends rotations to a representation for all Euclidean transformations. Furthermore, the rotation group provides a universal representation for all Lie groups. This was established in [2] by showing that any Lie algebra can be expressed as a bivector algebra. Since the Lie algebra describes the structure of the associated Lie group completely, any Lie group can be represented as rotation group.

The batch version of the problem was originally posed as the problem of estimating the attitude of satellites by Wahba in 1965 [3]. In psychometrics, it was presented as the orthogonal Procrustes problem [4]. It has been studied in various forms over the last few decades and finds application in many areas of computer vision [5, 6, 7], face recognition [8], robotics [9, 10], crystallography[11] and physics [12].

While the batch version of the problem is well understood, the online learning of rotations from vector instances is challenging since the manifold associated with the rotation group is a curved space and it is not possible to form updates that are linear combinations of rotations [13]. The set of rotations about the origin in $n$-dimensional Euclidean space forms a compact Lie group, $\mathbf{SO}(n)$, under the operation of composition. The manifold associated with the $n$-dimensional rotation group is the unit sphere $\mathbf{S}^{n-1}$ in $n$ dimensional Euclidean space.

### 1.1  Related Work

The online version of learning rotations was posed as an open problem by Smith and Warmuth [13]. Online learning algorithms were recently presented for some matrix groups. In [14], an online algorithm was proposed for learning density matrix parameters and was extended in [15] to the problem of learning subspaces of low rank. However, the extension of these algorithms to learning rotations will require repeated projection and approximation [13]. Adaptive algorithms were also

studied in [16] for optimization under unitary matrix constraint. The proposed methods are steepest descent methods on Riemannian manifolds.

## 1.2 Our Approach

This paper presents an online algorithm for learning rotations that utilizes the Bregman matrix divergence with respect to the quantum relative entropy (also known as *von Neumann divergence*) as a distance measure between two rotation matrices. The resulting algorithm has matrix-exponentiated gradient (MEG) updates [14]. The key ingredients of our approach are (a) von Neumann Divergence between rotation matrices [17], (b) squared error loss function and (c) matrix exponentiated gradient (MEG) updates.

Any Lie group is also a smooth manifold and the updates in the proposed algorithm have an intuitive interpretation in terms of the differential topology of the associated manifold. We also utilize various elementary Lie algebra concepts to provide intuitive interpretation of the updates. The development in the paper closely follows that of the matrix exponentiated gradient (MEG) updates in [14] for density matrix parameters. The form of the updates are similar to steepest descent methods of [16], but are derived for learning rotations from vector instances using an information-theoretic approach. The MEG updates are reduced to a quadratic form in the Lie algebra element corresponding to the gradient of loss function on the rotation group.

The paper is organized as follows. The problem is formulated in Section 2. Section 3 presents mathematical preliminaries in differential geometry and Bregman matrix divergence. The matrix exponentiated gradient updates are developed in Section 4. The MEG updates are simplified in Section 5. Experimental results are discussed in Section 6.

## 2 Problem Statement

Let $\mathbf{x}_t$ be a stream of instances of $n$-dimensional unit vectors. Let $\mathbf{R}_*$ be an unknown $n \times n$ rotation matrix that acts on $\mathbf{x}_t$ to give the rotated vector $\mathbf{y}_t = \mathbf{R}_* \mathbf{x}_t$. The matrix $\hat{\mathbf{R}}_t$ denotes the estimate of $\mathbf{R}_*$ at instance $t$ and $\hat{\mathbf{y}}_t = \hat{\mathbf{R}}_t \, \mathbf{x}_t$ represents the prediction for the rotated vector $\mathbf{y}_t$. The loss incurred due to error in prediction is $L_t(\hat{\mathbf{R}}_t) = d(\hat{\mathbf{y}}_t, \mathbf{y}_t)$, where $d(\cdot, \cdot)$ is a distance function. The estimate of the rotation needs to be updated based on the loss incurred at every instance and the objective is to develop an algorithm for learning $\mathbf{R}_*$ that has a bounded regret.

We seek adaptive updates that solve the following optimization problem at each step,

$$\hat{\mathbf{R}}_{t+1} = \arg \min_{\mathbf{R}} \Delta_F(\mathbf{R}, \hat{\mathbf{R}}_t) + \eta \, L_t(\mathbf{R}), \tag{1}$$

where $\hat{\mathbf{R}}_t$ is the estimated rotation matrix at instance $t$, $\eta$ is the learning rate or the step-size and $\Delta_F$ is a matrix divergence that measures the discrepancy between matrices. This is a typical problem formulation in online learning where the objective comprises a loss function and a divergence term. The parameter $\eta$ balances the trade-off between the two conflicting goals at each update: incurring small loss on the new data versus confidence in the estimate from the previously observed data. Minimizing the weighted objective therefore results in smooth updates as well as minimizes the loss function.

In this paper, the updates are smoothed using the von Neumann divergence which is defined for matrices as

$$\Delta_F(\mathbf{R}, \hat{\mathbf{R}}_t) = \text{tr}(\mathbf{R} \log \mathbf{R} - \mathbf{R} \log \hat{\mathbf{R}}_t - \mathbf{R} + \hat{\mathbf{R}}_t), \tag{2}$$

where $\text{tr}(\mathbf{A})$ is the trace of the matrix $\mathbf{A}$. The search is over all $\mathbf{R} \in \mathbf{SO}(n)$, i.e. over all $n \times n$ matrices such that $\mathbf{R}^T \mathbf{R} = \mathbf{I}$, $\mathbf{R}\mathbf{R}^T = \mathbf{I}$ and $\det(\mathbf{R}) = 1$.

## 3 Mathematical Preliminaries

This section reviews some basic definitions and concepts in linear algebra and differential geometry that are utilized for the development of the updates in the next section.

## 3.1 Matrix Calculus

Given a real-valued matrix function $F : \mathbb{R}^{n \times n} \to \mathbb{R}$, the gradient of the function with respect to the matrix $\mathbf{R} \in \mathbb{R}^{n \times n}$ is defined to be the matrix [18],

$$\nabla_{\mathbf{R}} F(\mathbf{R}) = \begin{pmatrix} \frac{\partial F}{\partial \mathbf{R}_{11}} & \cdots & \frac{\partial F}{\partial \mathbf{R}_{1n}} \\ \vdots & \ddots & \vdots \\ \frac{\partial F}{\partial \mathbf{R}_{n1}} & \cdots & \frac{\partial F}{\partial \mathbf{R}_{nn}} \end{pmatrix}. \tag{3}$$

Some of the matrix derivatives that are used later in the paper are following: for a constant matrix $\Gamma \in \mathbb{R}^{n \times n}$,

1. $\nabla_{\mathbf{R}} \operatorname{tr}(\Gamma \mathbf{R} \mathbf{R}^T) = (\Gamma + \Gamma^T)\mathbf{R}$,
2. $\nabla_{\mathbf{R}} \det(\mathbf{R}) = \det(\mathbf{R})(\mathbf{R}^{-1})^T$,
3. $\nabla_{\mathbf{R}} (\mathbf{y} - \mathbf{R}\mathbf{x})^T (\mathbf{y} - \mathbf{R}\mathbf{x}) = -2(\mathbf{y} - \mathbf{R}\mathbf{x})\mathbf{x}^T$.

A related concept in differential geometry is that of the space of vectors tangent to a group at the identity element of the group. This is defined to be the Lie algebra associated with the group. It is a convenient way of describing the infinitesimal structure of a topological group about the identity element and completely determines the associated group. The utility of the Lie algebra is due to the fact that it is a vector space and thus it is much easier to work with it than with the linear group.

A real $n \times n$ matrix $\mathbf{A}$ is in the Lie algebra of the rotation group $\mathbf{SO}(n)$ if and only if it is a skew-symmetric matrix (i.e. $\mathbf{A}^T = -\mathbf{A}$). Furthermore, for any matrix $\mathbf{A}$ in the Lie algebra of $\mathbf{SO}(n)$, $\exp(\eta \mathbf{A})$ is a one-parameter subgroup of the rotation group, parametrized by $\eta \in \mathbb{R}$ [19].

The matrix exponential and logarithm play an important role in relating a matrix Lie group $G$ and the associated Lie algebra $\mathfrak{g}$. The exponential of a matrix $\mathbf{R} \in \mathbb{R}^{n \times n}$ is given by the following series,

$$\exp(\mathbf{R}) = \mathbf{I} + \mathbf{R} + \frac{1}{2!}\mathbf{R}^2 + \frac{1}{3!}\mathbf{R}^2 + \cdots \tag{4}$$

Given an element $\mathbf{A} \in \mathfrak{g}$, the matrix exponential $\exp(\mathbf{A})$ is the corresponding element in the group. The matrix logarithm $\log(\mathbf{R})$ is defined to be the inverse of the matrix exponential: it maps from the Lie group $G$ into the Lie algebra $\mathfrak{g}$. The matrix logarithm is a well-defined map since the exponential map is a local diffeomorphism between a neighborhood of the zero matrix and a neighborhood of the identity matrix [19, 20].

## 3.2 Riemannian Gradient

Consider a real-valued differentiable function, $L_t : \mathbf{SO}(n) \to \mathbb{R}$, defined on the rotation group. The Riemannian gradient $\tilde{\nabla}_{\mathbf{R}} L_t$ of the function $L_t$ on the Lie group $\mathbf{SO}(n)$ evaluated at the rotation matrix $\mathbf{R}$ and translated to the identity (to get a Lie algebra element) is given as [16]

$$\tilde{\nabla}_{\mathbf{R}} L_t = \nabla_{\mathbf{R}} L_t \, \mathbf{R}^T - \mathbf{R} \, \nabla_{\mathbf{R}}^T L_t, \tag{5}$$

where $\nabla_{\mathbf{R}} L_t$ is the matrix derivative of the cost function in the Euclidean space defined in (3) at matrix $\mathbf{R}$.

## 3.3 Von Neumann Divergence

In any online learning problem, the choice of divergence between the parameters dictates the resulting updates. This paper utilizes the von Neumann divergence which is a special case of the Bregman divergence and measures discrepancy between two matrices.

Let $F$ be convex differentiable function defined on a subset of $\mathbb{R}^{n \times n}$ with the gradient $f(\mathbf{R}) = \nabla_{\mathbf{R}} F(\mathbf{R})$. The Bregman divergence between two matrices $\mathbf{R}_1$ and $\mathbf{R}_2$ is defined as

$$\Delta_F(\mathbf{R}_1, \mathbf{R}_2) := F(\mathbf{R}_1) - F(\mathbf{R}_2) - \operatorname{tr}((\mathbf{R}_1 - \mathbf{R}_2)f(\mathbf{R}_2)^T). \tag{6}$$

The gradient of Bregman divergence with respect to $\mathbf{R}_1$ is given as,

$$\nabla_{\mathbf{R}_1} \Delta_F(\mathbf{R}_1, \mathbf{R}_2) = f(\mathbf{R}_1) - f(\mathbf{R}_2). \tag{7}$$

Choosing the function $F$ in the definition of Bregman divergence to be the von Neumann entropy, given as $F(\mathbf{R}) = \text{tr}(\mathbf{R} \log \mathbf{R} - \mathbf{R}))$, obtain the von Neumann divergence [14, 17]:

$$\Delta_F(\mathbf{R}_1, \mathbf{R}_2) = \text{Tr}(\mathbf{R}_1 \log \mathbf{R}_1 - \mathbf{R}_1 \log \mathbf{R}_2 - \mathbf{R}_1 + \mathbf{R}_2). \tag{8}$$

Finally, the gradient of the von Neumann entropy was shown to be $f(\mathbf{R}) = \nabla_\mathbf{R} F(\mathbf{R}) = \log \mathbf{R}$ in [14]. Consequently, the gradient of the von Neumann divergence can be expressed as

$$\nabla_{\mathbf{R}_1} \Delta_F(\mathbf{R}_1, \mathbf{R}_2) = \log(\mathbf{R}_1) - \log(\mathbf{R}_2). \tag{9}$$

# 4 Online Algorithm

The problem of online learning of rotations can be expressed as the optimization problem

$$\hat{\mathbf{R}}_{t+1} = \underset{\mathbf{R}}{\arg\min} \quad \Delta_F(\mathbf{R}, \hat{\mathbf{R}}_t) + \eta L_t(\mathbf{R})$$
$$\text{s.t.} \quad \mathbf{R}^T \mathbf{R} = \mathbf{I}, \ \mathbf{R}\mathbf{R}^T = \mathbf{I}$$
$$\det(\mathbf{R}) = 1 \tag{10}$$

where $\hat{\mathbf{R}}_t$ is the estimate of the rotation matrix at time instance $t$ and $L_t$ is the loss incurred in the prediction of $\mathbf{y}_t$. The proposed adaptive updates are matrix exponentiated gradient (MEG) updates given as

$$\hat{\mathbf{R}}_{t+1} = \exp\left( \log \hat{\mathbf{R}}_t - \eta \, \mathbf{skew}\left( \hat{\mathbf{R}}_t^T \, \nabla_\mathbf{R} L_t(\hat{\mathbf{R}}_t) \right) \right), \tag{11}$$

where $\nabla_\mathbf{R} L_t(\hat{\mathbf{R}}_t)$ is the gradient of the cost function in the Euclidean space with respect to the rotation matrix $\mathbf{R}$ and $\mathbf{skew}\,(\cdot)$ is the skew-symmetrization operator on the matrices, $\mathbf{skew}\,(\mathbf{A}) = \mathbf{A} - \mathbf{A}^T$. The updates seem intuitive, given the following elementary facts about the Lie algebraic structure of the rotation group: (a) the gradient of loss function gives geodesic direction and velocity vector on the unit sphere, (b) a skew-symmetric matrix is an element of Lie algebra [19, 20], (c) the matrix logarithm maps a rotation matrix to the corresponding Lie algebra element, (d) composition of two elements of Lie algebra yields another Lie algebra element and (e) the matrix exponential maps a Lie algebra element to corresponding rotation matrix.

The loss function is defined to be the squared error loss function and therefore the gradient of the loss function is given by the matrix $\nabla_\mathbf{R} L_t(\hat{\mathbf{R}}_t) = 2(\hat{\mathbf{y}}_t - \mathbf{y}_t)\mathbf{x}_t^T$. This results in the online updates

$$\begin{aligned}
\hat{\mathbf{R}}_{t+1} &= \exp\left( \log \hat{\mathbf{R}}_t - 2\eta \, \mathbf{skew}\left( \hat{\mathbf{R}}_t^T (\hat{\mathbf{y}}_t - \mathbf{y}_t)\mathbf{x}_t^T \right) \right), \\
&= \hat{\mathbf{R}}_t \exp\left( -2\eta \, \mathbf{skew}\left( \hat{\mathbf{R}}_t^T (\hat{\mathbf{y}}_t - \mathbf{y}_t)\mathbf{x}_t^T \right) \right).
\end{aligned} \tag{12}$$

## 4.1 Updates Motivated by von-Neumann Divergence

The optimization problem in (10) is solved using the method of Lagrange multipliers. First observe that the constraints $\mathbf{R}^T \mathbf{R} = \mathbf{I}$ and $\mathbf{R}\mathbf{R}^T = \mathbf{I}$ are redundant since one implies the other. Introducing the Lagrangian multiplier matrix $\Gamma$ for the orthonormality constraint and Lagrangian multiplier $\lambda$ for the unity determinant constraint, the objective function can be written as

$$\mathcal{J}(\mathbf{R}, \Gamma, \lambda) = \Delta_F(\mathbf{R}, \hat{\mathbf{R}}_t) + \eta L_t(\mathbf{R}) + \text{tr}(\Gamma(\mathbf{R}\mathbf{R}^T - \mathbf{I})) + \lambda(\det(\mathbf{R}) - 1). \tag{13}$$

Taking the gradient on both sides of equation with respect to the matrix $\mathbf{R}$, get

$$\begin{aligned}
\nabla_\mathbf{R} \, \mathcal{J}(\mathbf{R}, \Gamma, \lambda) &= \nabla_\mathbf{R} \, \Delta_F(\mathbf{R}, \hat{\mathbf{R}}_t) + \eta \tilde{\nabla}_\mathbf{R} \, L_t(\mathbf{R}) \\
&\quad + (\Gamma + \Gamma^T)\mathbf{R} + \lambda \det(\mathbf{R})(\mathbf{R}^{-1})^T,
\end{aligned} \tag{14}$$

using the matrix derivatives from Section 3.1 and the Riemannian gradient for the loss function from eqn. (5). Putting $\nabla_{\mathbf{R}} \, \mathcal{J}(\mathbf{R}, \Gamma, \lambda) = 0$ and using the fact that $\nabla_{\mathbf{R}} \Delta_F(\mathbf{R}, \hat{\mathbf{R}}_t) = f(\mathbf{R}) - f(\hat{\mathbf{R}}_t)$, get

$$0 \;=\; f(\mathbf{R}) - f(\hat{\mathbf{R}}_t) + \eta \, \mathbf{skew}\left(\hat{\mathbf{R}}_t^T \nabla_{\mathbf{R}} L_t(\mathbf{R})\right) + (\Gamma + \Gamma^T)\mathbf{R} + \lambda \det(\mathbf{R})(\mathbf{R}^{-1})^T. \quad (15)$$

Given that $f$ is a bijective map, write

$$\mathbf{R} \;=\; f^{-1}\left(f(\hat{\mathbf{R}}_t) - \eta \, \mathbf{skew}\left(\hat{\mathbf{R}}_t^T \nabla_{\mathbf{R}} L_t(\mathbf{R})\right) - (\Gamma + \Gamma^T)\mathbf{R} - \lambda \det(\mathbf{R})(\mathbf{R}^{-1})^T\right). \quad (16)$$

Since the objective is convex, it is sufficient to produce a choice of Lagrange multipliers that enforces the rotation constraint. Choosing $\lambda = \det(\mathbf{R})^{-1}$ and $\Gamma = -(1/2)\left(\mathbf{R}^{-1}\right)^T \mathbf{R}^{-1}$ yields the following implicit update

$$\hat{\mathbf{R}}_{t+1} = \; \mathbf{exp}\left(\,\log \hat{\mathbf{R}}_t - \eta \, \mathbf{skew}\left(\hat{\mathbf{R}}_t^T \, \nabla_{\mathbf{R}} L_t(\hat{\mathbf{R}}_{t+1})\right)\right). \quad (17)$$

As noted by Tsuda et. al. in [14], the implicit updates of the form above are usually not solvable in closed form. However, by approximating $\nabla_{\mathbf{R}} L_t(\hat{\mathbf{R}}_{t+1})$ with $\nabla_{\mathbf{R}} L_t(\hat{\mathbf{R}}_t)$ (as in [21, 14]), we obtain an explicit update

$$\hat{\mathbf{R}}_{t+1} = \; \mathbf{exp}\left(\,\log \hat{\mathbf{R}}_t - \eta \, \mathbf{skew}\left(\hat{\mathbf{R}}_t^T \, \nabla_{\mathbf{R}} L_t(\hat{\mathbf{R}}_t)\right)\right). \quad (18)$$

The next result ensures the closure property for the matrix exponentiated gradient updates in the equation above. In other words, the estimates for the rotation matrix do not steer away from the manifold associated with the rotation group. Therefore, if $\hat{\mathbf{R}}_0 \in \mathbf{SO}(n)$ then $\hat{\mathbf{R}}_{t+1} \in \mathbf{SO}(n)$.

**Lemma 1.** *If $\hat{\mathbf{R}}_t \in \mathbf{SO}(n)$ then $\hat{\mathbf{R}}_{t+1}$ given by the updates in* (18) *is a rotation matrix in* $\mathbf{SO}(n)$.

*Proof.* Using the properties of matrix logarithm and matrix exponential, express (18) as

$$\hat{\mathbf{R}}_{t+1} = \hat{\mathbf{R}}_t \, \mathbf{exp}(-\eta \mathbf{S}), \quad (19)$$

where $\mathbf{S} = \hat{\mathbf{R}}_t^T \, \nabla_{\mathbf{R}} L_t(\mathbf{R}) - \nabla_{\mathbf{R}}^T L_t(\mathbf{R}) \, \hat{\mathbf{R}}_t$ is an $n \times n$ dimensional skew-symmetric matrix with trace zero. Then

$$\begin{aligned}
\hat{\mathbf{R}}_{t+1}^T \, \hat{\mathbf{R}}_{t+1} &= \left(\hat{\mathbf{R}}_t \, e^{-\eta \mathbf{S}}\right)^T \left(\hat{\mathbf{R}}_t \, e^{-\eta \mathbf{S}}\right), \\
&= \left(e^{-\eta \mathbf{S}}\right)^T \hat{\mathbf{R}}_t^T \, \hat{\mathbf{R}}_t \left(e^{-\eta \mathbf{S}}\right), \\
&= \left(e^{-\eta \mathbf{S}}\right)^T \left(e^{-\eta \mathbf{S}}\right), \\
&= e^{\eta(-\mathbf{S}^T - \mathbf{S})} = e^{\eta(\mathbf{S} - \mathbf{S})} = \mathbf{I},
\end{aligned}$$

where we used the facts that $\hat{\mathbf{R}}_t \in \mathbf{SO}(n)$, $\left(e^S\right)^T = e^{S^T}$, $\mathbf{S}^T = -\mathbf{S}$ and that $e^0 = \mathbf{I}$. Similarly, $\hat{\mathbf{R}}_{t+1} \, \hat{\mathbf{R}}_{t+1}^T = \mathbf{I}$. Finally, note that

$$\det(\hat{\mathbf{R}}_{t+1}) = \det(\hat{\mathbf{R}}_t \, e^{-\eta \mathbf{S}}) = \det(\hat{\mathbf{R}}_t) \cdot \det(e^{-\eta \mathbf{S}}) = e^{-\eta \, \mathrm{Tr}\,(\mathbf{S})},$$

since determinant of exponential of a matrix is equal to the exponential of the trace of the matrix. And since $\mathbf{S}$ is a trace zero matrix, $\det(\hat{\mathbf{R}}_{t+1}) = 1$. $\qquad\square$

## 4.2 Differential Geometrical Interpretation

The resulting updates in (18) have nice interpretation in terms of the differential geometry of the rotation group. The gradient of the cost function, $\nabla_{\mathbf{R}} L_t(\hat{\mathbf{R}}_t)$, in the Euclidean space gives a tangent direction at the current estimate of the rotation matrix. The Riemannian gradient is computed as $\nabla_{\mathbf{R}} L_t(\hat{\mathbf{R}}_t) - \hat{\mathbf{R}}_t \, \nabla_{\mathbf{R}}^T L_t(\hat{\mathbf{R}}_t) \, \hat{\mathbf{R}}_t$. The Riemannian gradient at the identity element of the group is obtained by de-rotation by $\hat{\mathbf{R}}_t$, giving $\tilde{\nabla}_{\mathbf{R}} L_t(\hat{\mathbf{R}}_t)$, as in (5). The gradient corresponds to an element of the Lie algebra, $\mathfrak{so}(n)$, of the rotation group. The exponential map gives the corresponding rotation matrix which is the multiplicative update to the estimate of the rotation matrix at the previous instance.

## 5   Complexity Reduction of MEG Updates

The matrix exponentiated gradient updates ensure that the estimates for the rotation matrix stay on the manifold associated with the rotation group at each iteration. However, with the matrix exponentiation at each step, the updates are computationally intensive and in fact the computational complexity of the updates is comparable to other approaches that would require repeated approximation and projection on to the manifold. This section discusses a fundamental complexity reduction result to establish a simpler update by exploiting the eigen-structure of the update matrix. First observe that the matrix in the exponential in eqn. (12) (for the case of squared error loss function) can be written as

$$
\begin{aligned}
\mathbf{S} &= -2\eta \; \mathbf{skew}\left(\hat{\mathbf{R}}_t^T(\hat{\mathbf{y}}_t - \mathbf{y}_t)\mathbf{x}_t^T\right), \\
&= -2\eta \; \mathbf{skew}\left(\hat{\mathbf{R}}_t^T(\hat{\mathbf{R}}_t\mathbf{x}_t - \mathbf{R}_*\mathbf{x}_t)\mathbf{x}_t^T\right), \\
&= -2\eta \; \mathbf{skew}\left(\mathbf{x}_t\mathbf{x}_t^T - \hat{\mathbf{R}}_t^T\mathbf{R}_*\mathbf{x}_t\mathbf{x}_t^T\right), \\
&= 2\eta \left(\hat{\mathbf{R}}_t^T\mathbf{R}_*\mathbf{x}_t\mathbf{x}_t^T - \mathbf{x}_t\mathbf{x}_t^T\mathbf{R}_*^T\hat{\mathbf{R}}_t\right), \\
&= \mathbf{A}^T\mathbf{X} - \mathbf{X}\mathbf{A}, &(20)
\end{aligned}
$$

where $\mathbf{X} \equiv \mathbf{x}_t\mathbf{x}_t^T$ and $\mathbf{A} \equiv 2\eta\mathbf{R}_*^T\hat{\mathbf{R}}_t$. Each term in the matrix $\mathbf{S}$ is a rank-one matrix (due to pre and post-multiplication with $\mathbf{x}_t\mathbf{x}_t^T$, respectively). Thus $\mathbf{S}$ is at most rank-two. Since $\mathbf{S}$ is skew-symmetric, it has (at most) two eigenvalues in a complex conjugate pair $\pm j\lambda$ (and $n-2$ zero eigenvalues) [22], which allows the following simplification.

**Lemma 2.** *The matrix exponentiated gradient updates in eqn. (12) are equivalent to the following updates,*

$$
\hat{\mathbf{R}}_{t+1} = \hat{\mathbf{R}}_t \left(\mathbf{I} + \frac{\sin(\lambda)}{\lambda}\mathbf{S} + \frac{1 - \cos(\lambda)}{\lambda^2}\mathbf{S}^2\right), \tag{21}
$$

*where $\lambda = 2\eta\sqrt{1 - \left(\mathbf{y}_t^T\hat{\mathbf{y}}_t\right)^2}$ and $\mathbf{S}$ is the skew-symmetric matrix given in eqn. (20) with eigenvalues $\pm j\lambda$.*

Note that $\mathbf{y}_t, \hat{\mathbf{y}}_t$ are unit vectors in $\mathbb{R}^n$ and therefore $\lambda$ is real-valued. The proof of the complexity reduction follows easily from a generalization of the Rodrigues' formula for computing matrix exponentials for skew-symmetric matrix. The proof is not presented here due to space constraints but the interested reader is referred to [23, 24]. Owing to the result above the matrix exponential reduces to a simple quadratic form involving an element from the Lie algebra of the rotation group. The pseudocode is given in Algorithm 1.

---

Choose $\eta$
Initialize $\mathbf{R}_1 = \mathbf{I}$
**for** $t = 1, 2, \ldots$ **do**

> Obtain an instance of unit vector $\mathbf{x}_t \in \mathbb{R}^n$;
>
> Predict the rotated vector $\hat{\mathbf{y}}_t = \hat{\mathbf{R}}_t \, \mathbf{x}_t$;
>
> Receive the true rotated vector $\mathbf{y}_t = \mathbf{R}_* \, \mathbf{x}_t$;
>
> Incur the loss $L_t(\hat{\mathbf{R}}_t) = |\mathbf{y}_t - \hat{\mathbf{y}}_t|^2$;
>
> Compute the matrix $\mathbf{S} = 2\eta \left(\hat{\mathbf{R}}_t^T\mathbf{y}_t\mathbf{x}_t^T - \mathbf{x}_t\mathbf{y}_t^T\hat{\mathbf{R}}_t\right)$;
>
> Compute the eigenvalues $\lambda = 2\eta\sqrt{1 - \left(\mathbf{y}_t^T\hat{\mathbf{y}}_t\right)^2}$;
>
> Update the rotation matrix $\hat{\mathbf{R}}_{t+1} = \hat{\mathbf{R}}_t \left(\mathbf{I} + \frac{\sin(\lambda)}{\lambda}\mathbf{S} + \frac{1 - \cos(\lambda)}{\lambda^2}\mathbf{S}^2\right)$

**end**

> **Algorithm 1**: Pseudocode for Learning rotations using Matrix Exponentiated Gradient updates

---

# 6  Experimental Results

This section presents experimental results with the proposed algorithm for online learning of rotations. The performance of the algorithm is evaluated in terms of the Frobenius norm of the difference of the true rotation matrix and the estimate. Figure 1 shows the error plot with respect to time. The unknown rotation is a $12 \times 12$ dimensional matrix and changes randomly every 200 instances. The trajectories are averaged over 1000 random simulations. It is clear from the plot that the estimation error decays rapidly to zero and estimates of the rotation matrices are exact.

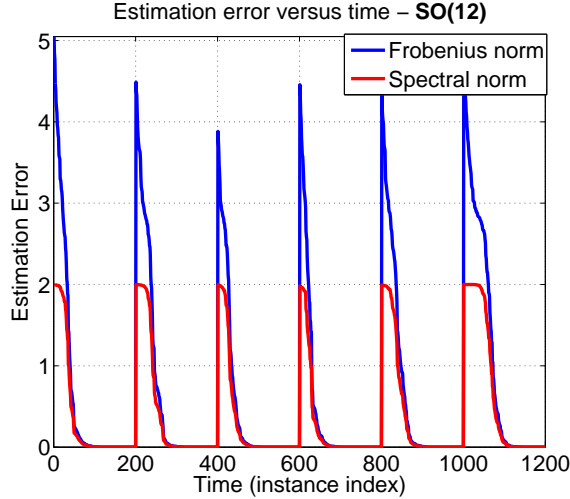

Figure 1: Online learning of rotations: Estimate of unknown rotation is updated every time new instance of rotation is observed. The true rotation matrix is randomly changing at regular interval (N=200). The error in Frobenius norm is plotted against the instance index.

The online algorithm is also found robust to small amount of additive white Gaussian noise in observations of the true rotated vectors, i.e. the observations are now given as $\mathbf{y}_t = \mathbf{R}_* \mathbf{x}_t + \alpha \, \mathbf{w}_t$, where $\alpha$ determines the signal to noise ratio. The performance of the algorithm is studied with various noisy conditions. Figure 2 shows error plots with respect to time for various noisy conditions in $\mathbb{R}^{20}$. The Frobenius norm error decays quickly to a noise floor determined by the SNR as well as the step size $\eta$. In the simulations in Fig. 2 the step size was decreased gradually over time. It is not clear immediately how to pick the optimal step size but a classic step size adaptation rule or Armijo rule may be followed [25, 16].

The tracking performance of the online algorithm is compared with the batch version. In Figure 3, the unknown rotation $\mathbf{R}_* \in \mathbf{SO}(30)$ changes slightly after every 30 instances. The smoothly changing rotation is induced by composing $\mathbf{R}_*$ matrix with a matrix $\mathbf{R}_\delta$ every thirty iterations. The matrix $\mathbf{R}_\delta$ is composed of $3 \times 3$ block-diagonal matrices, each corresponding to rotation about the $X$-axis in 3D space by $\pi/360$ radians. The batch version stores the last 30 instances in an $30 \times 30$ matrix $\mathbf{X}$ and corresponding rotated vectors in matrix $\mathbf{Y}$. The estimate of the unknown rotation is given as $\mathbf{Y}\mathbf{X}^{-1}$. The batch version achieves zero error only at time instances when all the data in $\mathbf{X}, \mathbf{Y}$ correspond to the same rotation whereas the online version consistently achieves a low error and tracks the changing rotation.

It is clear from the simulations that the Frobenius norm decreases at each iteration. It is easy to show this global stability of the updates proposed here in noise-free scenario [24]. The proposed algorithm was also applied to learning and tracking the rotations of 3D objects. Videos showing experimental results with the 3D Stanford bunny [26] are posted online at [27].

# 7  Conclusion

In this paper, we have presented an online algorithm for learning rotations. The algorithm was motivated using the von Neumann divergence and squared error loss function and the updates were

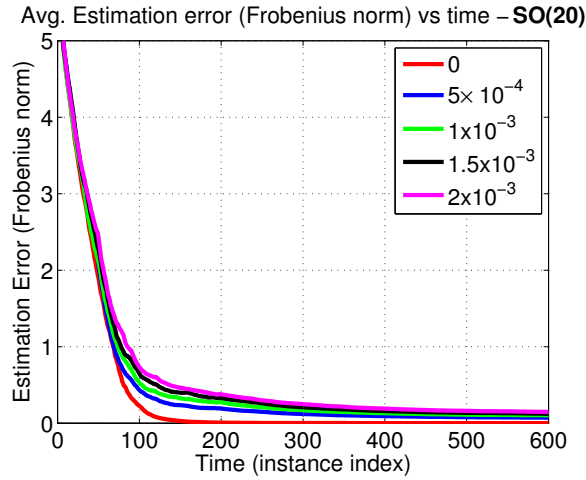

Figure 2: Average error plotted against instance index for various noise levels.

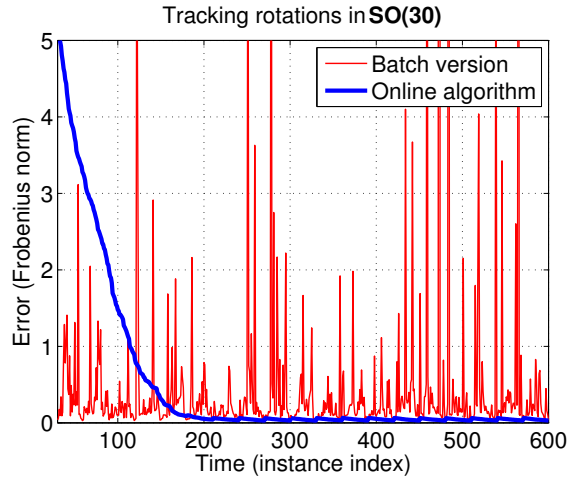

Figure 3: Comparing the performance of tracking rotations for the batch version versus the online algorithm. The rotation matrix changes smoothly every $M = 30$ instances.

developed in the Lie algebra of the rotation group. The resulting matrix exponentiated gradient updates were reduced to a simple quadratic form. The estimation performance of the proposed algorithm was studied under various scenarios. Some of the future directions include identifying alternative loss functions that exploit the spherical geometry as well as identifying regret bounds for the proposed updates.

**Acknowledgements:** The author would like to thank W. A. Sethares, M. R. Gupta and A. B. Frigyik for helpful discussions and feedback on early drafts of the paper.

# References

[1] Rich Wareham, Jonathan Cameron, and Joan Lasenby, "Applications of conformal geometric algebra in computer vision and graphics," in *IWMM/GIAE*, 2004, pp. 329–349.

[2] C. Doran, D. Hestenes, F. Sommen, and N. Van Acker, "Lie groups as spin groups," *Journal of Mathematical Physics*, vol. 34, no. 8, pp. 36423669, August 1993.

[3] Grace Wahba, "Problem 65-1, a least squares estimate of satellite attitude," *SIAM Review*, vol. 7, no. 3, July 1965.

[4] P. Schonemann, "A generalized solution of the orthogonal Procrustes problem," *Psychometrika*, vol. 31, no. 1, pp. 3642–3669, March 1966.

[5] P. Besl and N. McKay, "A method for registration of 3D shapes," . *IEEE Trans. on Pattern Analysis and Machine Intelligence*, vol. 14, pp. 239–256, 1992.

[6] Hannes Edvardson and Örjan Smedby, "Compact and efficient 3D shape description through radial function approximation," *Computer Methods and Programs in Biomedicine*, vol. 72, no. 2, pp. 89–97, 2003.

[7] D.W. Eggert, A. Lorusso, and R.B. Fisher, "Estimating 3D rigid body transformations: a comparison of four major algorithms," *Machine Vision and Applications, Springer*, vol. 9, no. 5-6, Mar 1997.

[8] R. Sala Llonch, E. Kokiopoulou, I. Tosic, and P. Frossard, "3D face recognition with sparse spherical representations," *Preprint, Elsiever*, 2009.

[9] Ameesh Makadia and Kostas Daniilidis, "Rotation recovery from spherical images without correspondences," *IEEE Trans. Pattern Analysis Machine Intelligence*, vol. 28, no. 7, pp. 1170–1175, 2006.

[10] Raman Arora and Harish Parthasarathy, "Navigation using a spherical camera," in *International Conference on Pattern Recognition (ICPR)*, Tampa, Florida, Dec 2008.

[11] Philip R. Evans, "Rotations and rotation matrices," *Acta Cryst.*, vol. D57, pp. 1355–1359, 2001.

[12] Richard L. Liboff, *Introductory Quantum Mechanics*, Addison-Wesley, 2002.

[13] Adam Smith and Manfred Warmuth, "Learning rotations," in *Conference on Learning Theory (COLT)*, Finland, Jun 2008.

[14] Koji Tsuda, Gunnar Ratsch, and Manfred K Warmuth, "Matrix exponentiated gradient updates for on-line learning and Bregman projection," *Journal of Machine Learning Research*, vol. 6, Jun 2005.

[15] Manfred K Warmuth, "Winnowing subspaces," in *Proc. 24th Int. Conf. on Machine Learning*, 2007.

[16] T.E. Abrudan, J. Eriksson, and V. Koivunen, "Steepest descent algorithms for optimization under unitary matrix constraint," *Signal Processing, IEEE Transactions on*, vol. 56, no. 3, pp. 1134–1147, March 2008.

[17] M. A. Nielsen and I. L. Chuang, *Quantum Computation and Quantum Information*, Cambridge, 2000.

[18] Kaare Brandt Petersen and Michael Syskind Pedersen, "The matrix cookbook," `http://matrixcookbook.com`, November 14, 2008.

[19] Michael Artin, *Algebra*, Prentice Hall, 1991.

[20] John A. Thorpe, *Elementary topics in Differential Geometry*, Springer-Verlag, 1994.

[21] J. Kivinen andM. K.Warmuth, "Exponentiated gradient versus gradient descent for linear predictors," *Information and Computation*, vol. 132, no. 1, pp. 1–63, Jan 1997.

[22] L. J. Butler, *Applications of Matrix Theory to Approximation Theory*, MS Thesis, Texas Tech University, Aug. 1999.

[23] J. Gallier and D. Xu, "Computing exponentials of skew-symmetric matrices and logarithms of orthogonal matrices," *International Journal of Robotics and Automation*, vol. 17, no. 4, 2002.

[24] Raman Arora, *Group theoretical methods in signal processing: learning similarities, transformation and invariants*, Ph.D. thesis, University of Wisconsin-Madison, Madison, August 2009.

[25] E. Polak, *Optimization: Algorithms and Consistent Approximations*, Springer-Verlag, 1997.

[26] Stanford University Computer Graphics Laboratory, "The Stanford 3D scanning repository," `http://graphics.stanford.edu/data/`.

[27] Raman Arora, "Tracking rotations of 3D Stanford bunny," `http://www.cae.wisc.edu/~sethares/links/raman/LearnROT/vids.html`, 2009.

